# Controlling the Complexity of HMM Systems by Regularization

**Christoph Neukirchen, Gerhard Rigoll**
Department of Computer Science
Gerhard-Mercator-University Duisburg
47057 Duisburg, Germany
email: {chn,rigoll}@fb9-ti.uni-duisburg.de

## Abstract

This paper introduces a method for regularization of HMM systems that avoids parameter overfitting caused by insufficient training data. Regularization is done by augmenting the EM training method by a penalty term that favors simple and smooth HMM systems. The penalty term is constructed as a mixture model of negative exponential distributions that is assumed to generate the state dependent emission probabilities of the HMMs. This new method is the successful transfer of a well known regularization approach in neural networks to the HMM domain and can be interpreted as a generalization of traditional state-tying for HMM systems. The effect of regularization is demonstrated for continuous speech recognition tasks by improving overfitted triphone models and by speaker adaptation with limited training data.

## 1 Introduction

One general problem when constructing statistical pattern recognition systems is to ensure the capability to generalize well, i.e. the system must be able to classify data that is not contained in the training data set. Hence the classifier should learn the true underlying data distribution instead of overfitting to the few data examples seen during system training. One way to cope with the problem of overfitting is to balance the system's complexity and flexibility against the limited amount of data that is available for training.

In the neural network community it is well known that the amount of information used in system training that is required for a good generalization performance should be larger than the number of adjustable weights (Baum, 1989). A common method to train a large size neural network sufficiently well is to reduce the number of adjustable parameters either by removing those weights that seem to be less important (in (le Cun, 1990) the sensitivity of individual network weights is estimated by the second order gradient) or by sharing

the weights among many network connections (in (Lang, 1990) the connections that share identical weight values are determined in advance by using prior knowledge about invariances in the problem to be solved). A second approach to avoid overfitting in neural networks is to make use of regularization methods. Regularization adds an extra term to the training objective function that penalizes network complexity. The simplest regularization method is weight decay (Plaut, 1986) that assigns high penalties to large weights. A more complex regularization term is used in soft weight-sharing (Nowlan, 1992) by favoring neural network weights that fall into a finite set of small weight-clusters. The traditional neural weight sharing technique can be interpreted as a special case of soft weight-sharing regularization when the cluster variances tend towards zero.

In continuous speech recognition the Hidden Markov Model (HMM) method is common. When using detailed context-dependent triphone HMMs, the number of HMM-states and parameters to estimate in the state-dependent probability density functions (pdfs) is increasingly large and overfitting becomes a serious problem. The most common approach to balance the complexity of triphone HMM systems against the training data set is to reduce the number of parameters by tying, i.e. parameter sharing (Young, 1992). A popular sharing method is state-tying with selecting the HMM-states to be tied in advance, either by data-driven state-clustering based on a pdf-dependent distance metric (Young, 1993), or by constructing binary decision trees that incorporate higher phonetic knowledge (Bahl, 1991). In these methods, the number of state-clusters and the decision tree sizes, respectively, must be chosen adequately to match the training data size. However, a possible drawback of both methods is that two different states may be selected to be tied (and their pdfs are forced to be identical) although there is enough training data to estimate the different pdfs of both states sufficiently well. In the following, a method to reduce the complexity of general HMM systems based on a regularization term is presented. Due to its close relationship to the soft weight-sharing method for neural networks this novel approach can be interpreted as soft state-tying.

## 2   Maximum likelihood training in HMM systems

Traditionally, the method most commonly used to determine the set of adjustable parameters $\Theta$ in a HMM system is maximum likelihood (ML) estimation via the expectation maximization (EM) algorithm. If the training observation vector sequence is denoted as $X = (\mathbf{x}(1), \ldots, \mathbf{x}(T))$ and the corresponding HMM is denoted as $W$ the ML estimator is given by:

$$\hat{\theta}^{ML} = \underset{\theta}{\operatorname{argmax}} \left\{ \log p_{\Theta}(X|W) \right\} \tag{1}$$

In the following, the total number of different HMM states is given by $K$. The emission pdf of the $k$-th state is denoted as $b_k(\mathbf{x})$; for continuous HMMs $b_k(\mathbf{x})$ is a mixture of Gaussian pdfs most commonly; in the case of discrete HMMs the observation vector $\mathbf{x}$ is mapped by a vector quantizer (VQ) on the discrete VQ-label $\hat{m}(\mathbf{x})$ and the emission pdf is replaced by the discrete output probability $b_k(\hat{m})$. By the forward-backward algorithm the probabilistic state counts $\gamma_k(t)$ can be determined for each training observation and the log-likelihood over the training data can be decomposed into the auxiliary function $Q(\Theta)$ optimized in the EM steps (state transition probabilities are neglected here):

$$Q(\Theta) = \sum_{t=1}^{T} \sum_{k=1}^{K} \gamma_k(t) \cdot \log b_k(\mathbf{x}(t)) \tag{2}$$

Sometimes, the observation vector $\mathbf{x}$ is split up into several independent streams. If the total number of streams is given by $Z$, the features in the $z$-th stream comprise the subvector $\mathbf{x}^{(z)}$ and in the case of application of a VQ the corresponding VQ label is denoted as $\hat{m}^{(z)}(\mathbf{x}^{(z)})$.

The observation subvectors in different streams are assumed to be statistically independent thus the states' pdfs can be written as:

$$b_k(\mathbf{x}) = \prod_{z=1}^{Z} b_k^{(z)}(\mathbf{x}^{(z)}) \tag{3}$$

## 3  A complexity measure for HMM systems

When using regularization methods to train the HMM system, the traditional objective training function $Q(\Theta)$ is augmented by a complexity penalization term $\Omega$ and the new optimization problem becomes:

$$\hat{\theta}^{reg} = \operatorname*{argmax}_{\theta} \left\{ Q(\Theta) + \nu \cdot \Omega(\Theta) \right\} \tag{4}$$

Here, the regulizer term $\Omega$ should be small if the HMM system has high complexity and parameter overfitting becomes a problem; $\Omega$ should be large if the HMM-states' pdfs are shaped smoothly and system generalization works well. The constant $\nu \geq 0$ is a control parameter that adjusts the tradeoff between the pure ML solution and the smoothness of penalization. In Eqn. (4) the term $Q(\Theta)$ becomes larger the more data is used for training (which makes the ML estimation become more reliable) and the influence of the term $\nu \cdot \Omega$ gets less important, relatively.

The basic idea when constructing an expression for the regulizer $\Omega$ that favors smooth HMM systems is, that in the case of simple and smooth systems the state-dependent emission pdfs $b_k(\cdot)$ should fall into several groups of similar pdfs. This is in contrast to the traditional state-tying that forces identical pdfs in each group. In the following, these clusters of similar emission pdfs are described by a probabilistic mixture model. Each pdf is assumed to be generated by a mixture of $I$ different mixture components $p_i(\cdot)$. In this case the probability (-density) of generating the emission pdf $b_k(\cdot)$ is given by:

$$p(b_k(\cdot)) = \sum_{i=1}^{I} c_i \cdot p_i(b_k(\cdot)) \tag{5}$$

with the mixture weights $c_i$ that are constrained to $0 \leq c_i \leq 1$ and $1 = \sum_{i=1}^{I} c_i$. The $i$-th mixture component $p_i(\cdot)$ is used to model the $i$-th cluster of HMM-emission pdfs. Each cluster is represented by a prototype pdf that is denoted as $\beta_i(\cdot)$ for the $i$-th cluster; the distance (using a suitable metric) between a HMM emission pdf $b_k(\cdot)$ and the $i$-th prototype pdf is denoted as $D_i(b_k(\cdot))$. If these distances are small for all HMM emission probabilities there are several small clusters of emission probabilities and the regulizer term $\Omega$ should be large. Now, it is assumed that the distances follow a negative exponential distribution (with a deviation parameter $\lambda_i$), yielding an expression for the mixture components:

$$p_i(b_k(\cdot)) \sim \left( \prod_{z=1}^{Z} \lambda_{i,z} \right) \cdot \exp\left( -\sum_{z=1}^{Z} \lambda_{i,z} \cdot D_{i,z}(b_k^{(z)}(\cdot)) \right) \tag{6}$$

In Eqn. (6) the more general case of $Z$ independent streams is given. Hence, the HMM emission pdfs and the cluster prototype pdfs are split up into $Z$ different pdfs $b_k^{(z)}(\cdot)$ and $\beta_i^{(z)}(\cdot)$, respectively and the stream dependent distances $D_{i,z}$ and parameters $\lambda_{i,z}$ are used.

Now, for the regulizer term $\Omega$ the log-likelihood of the mixture model in Eqn. (5) over all emission pdfs in the HMM system can be used:

$$\Omega(\Theta) = \sum_{k=1}^{K} \log p(b_k(\cdot)) \tag{7}$$

## 4   Regularization example: discrete HMMs

As an example for parameter estimation in the regularization framework, a discrete HMM system with different VQs for each of the $Z$ streams is considered here: Each VQ subdivides the feature space into $J_z$ different partitions (i.e. the $z$-th codebook size is $J_z$) and the VQ-partition labels are denoted $m_j^{(z)}$. If the observation subvector $\mathbf{x}^{(z)}$ is in the $j$-th VQ-partition the VQ output is $\hat{m}^{(z)}(\mathbf{x}^{(z)}) = m_j^{(z)}$.

Since the discrete kind HMM output probabilities $b_k^{(z)}(\hat{m}^{(z)})$ are used here, the regulizer's prototypes are the discrete probabilities $\beta_i^{(z)}(m_j^{(z)})$. As a distance metric between the HMM emission probabilities and the prototype probabilities used in Eqn. (6) the asymmetric Kullback-Leibler divergence is applied:

$$D_{i,z}(b_k^{(z)}(\hat{m}^{(z)})) = -\sum_{j=1}^{J_z} \beta_i^{(z)}(m_j^{(z)}) \cdot \log \frac{b_k^{(z)}(m_j^{(z)})}{\beta_i^{(z)}(m_j^{(z)})} \tag{8}$$

### 4.1   Estimation of HMM parameters using regularization

The parameter set $\Theta$ of the HMM system to be estimated mainly consists of the discrete HMM emission probabilities (transition probabilities are not subject of regularization here). To get an iterative parameter estimation in the EM style, Eqn. (4) must be maximized; e.g. by setting the derivative of Eqn. (4) with respect to the HMM-parameter $b_k^{(z)}(m_j^{(z)})$ to zero and application of Lagrange multipliers with regard to the constraint $1 = \sum_{j=1}^{J_z} b_k^{(z)}(m_j^{(z)})$. This leads to a quite complex solution that can be only solved numerically.

The optimization problem can be simplified if the mixture in Eqn. (5) is replaced by the maximum approximation; i.e. only the maximum component in the sum is considered. The corresponding index of the maximum component is denoted $i^*$:

$$p(b_k(\cdot)) \approx c_{i^*} \cdot p_{i^*}(b_k(\cdot)) = \max_{1 \leq i \leq I} \{c_i \cdot p_i(b_k(\cdot))\} \tag{9}$$

In this simplified case the HMM parameter estimation is given by:

$$\hat{b}_k^{(z)}(m_j^{(z)}) = \frac{\dfrac{\nu}{\lambda_{i^*,z}} \cdot \beta_{i^*}^{(z)}(m_j^{(z)}) + \sum_{t=1}^{T} \gamma_k(t) \cdot \delta_{\hat{m}^{(z)}(t),m_j^{(z)}}}{\dfrac{\nu}{\lambda_{i^*,z}} + \sum_{t=1}^{T} \gamma_k(t)} \tag{10}$$

This is a weighted sum of the well known ML solution and the regulizer's prototype probability $\beta_{i^*}^{(z)}(\cdot)$ that is selected by the maximum search in Eqn. (9). The larger the value of the constant $\nu$, the stronger is the force that pushes the estimate of the HMM emission probability $\hat{b}_k^{(z)}(m_j^{(z)})$ towards the prototype probability $\beta_{i^*}^{(z)}(\cdot)$. The situation when $\nu$ tends towards infinity corresponds to the case of traditional state-tying, because all different states that fall into the same cluster $i^*$ make use of $\beta_{i^*}^{(z)}(\cdot)$ as emission probability in the $z$-th stream.

### 4.2   Estimation of regulizer parameters

The parameter set $\xi$ of the regulizer consists of the mixture weights $c_i$, the deviation parameters $\lambda_{i,z}$, and of the discrete prototype probabilities $\beta_i^{(z)}(m_j^{(z)})$ in the case of regulizing

discrete HMMs. These parameters can be set in advance by making use of prior knowledge; e.g. the prototype probabilities can be obtained from a simple HMM system that uses a small number of states. Alternatively, the regulizer's parameters can be estimated in a similar way as in (Nowlan, 1992) by maximizing Eqn. (7). Since there is no direct solution to this optimization problem, maximization must be performed in an EM-like iterative procedure that uses the HMM emission pdfs $b_k(\cdot)$ as training data for the mixture model and by increasing the following auxiliary function in each step:

$$
\begin{aligned}
R(\xi) &= \sum_{k=1}^{K} \sum_{i=1}^{I} P(i|b_k(\cdot)) \cdot \log p(i, b_k(\cdot)) \\
&= \sum_{k=1}^{K} \sum_{i=1}^{I} P(i|b_k(\cdot)) \cdot \log \left( c_i \cdot p_i(b_k(\cdot)) \right)
\end{aligned}
\tag{11}
$$

with the posterior probability used as weighting factor given by:

$$
P(i|b_k(\cdot)) = \frac{c_i \cdot p_i(b_k(\cdot))}{\sum_{l=1}^{I} c_l \cdot p_l(b_k(\cdot))}
\tag{12}
$$

Again, maximization of Eqn. (11) can be performed by setting the derivative of $R(\xi)$ with respect to the regulizer's parameters to zero under consideration of the constraints $1 = \sum_{i=1}^{I} c_i$ and $1 = \sum_{j=1}^{J_z} \beta_i^{(z)}(m_j^{(z)})$ by application of Lagrange multipliers. For the estimation of the regulizer parameters this yields:

$$
\hat{c}_i = \frac{1}{K} \cdot \sum_{k=1}^{K} P(i|b_k(\cdot))
\tag{13}
$$

$$
\hat{\lambda}_{i,z} = \frac{\sum_{k=1}^{K} P(i|b_k(\cdot))}{\sum_{k=1}^{K} D_{i,z}(b_k^{(z)}(\cdot)) \cdot P(i|b_k(\cdot))}
\tag{14}
$$

$$
\hat{\beta}_i^{(z)}(m_j^{(z)}) = \frac{\exp \left( \dfrac{\sum_{k=1}^{K} P(i|b_k(\cdot)) \cdot \log b_k^{(z)}(m_j^{(z)})}{\sum_{k=1}^{K} P(i|b_k(\cdot))} \right)}{\sum_{l=1}^{I} \exp \left( \dfrac{\sum_{k=1}^{K} P(l|b_k(\cdot)) \cdot \log b_k^{(z)}(m_j^{(z)})}{\sum_{k=1}^{K} P(l|b_k(\cdot))} \right)}
\tag{15}
$$

The estimate $\hat{c}_i$ can be interpreted as the average probability that a HMM emission probability falls into the $i$-th mixture cluster; $\hat{\lambda}_{i,z}$ is the inverse of the weighted average distance between the emission probabilities and the prototype probability $\beta_i^{(z)}(\cdot)$. The estimate $\hat{\beta}_i^{(z)}(m_j^{(z)})$ is the average probability over all emission probabilities for the VQ-label $m_j^{(z)}$ weighted in the log-domain.

If the Euclidean distance between the discrete probabilities is used instead of Eqn. (8) to measure the differences between the HMM emission probabilities and the prototypes

$$
D_{i,z}(b_k^{(z)}(\hat{m}^{(z)})) = \sum_{j=1}^{J_z} \left( \beta_i^{(z)}(m_j^{(z)}) - b_k^{(z)}(m_j^{(z)}) \right)^2
\tag{16}
$$

the estimate of the prototype probabilities is given by the average of the HMM probabilities weighted in the original space:

$$
\hat{\beta}_i^{(z)}(m_j^{(z)}) = \frac{\sum_{k=1}^{K} P(i|b_k(\cdot)) \cdot b_k^{(z)}(m_j^{(z)})}{\sum_{k=1}^{K} P(i|b_k(\cdot))}
\tag{17}
$$

# 5   Experimental results

To investigate the performance of the regularization methods described above a HMM speech recognition system for the speaker-independent resource management (RM) continuous speech task is built up. For training 3990 sentences from 109 different speakers are used. Recognition results are given as word error rates averaged over the official DARPA RM test sets feb'89, oct'89, feb'91 and sep'92, consisting of 1200 sentences from 40 different speakers, totally. Recognition is done via a beam search guided Viterbi decoder using the DARPA RM word pair grammar (perplexity: 60).

As acoustic features every 10 ms 12 MFCC coefficients and the relative signal power are extracted from the speech signal along with the dynamic $\Delta$- and $\Delta\Delta$-features, comprising 39 features per frame. The HMM system makes use of standard 3-state discrete probability phonetic models. Four different neural networks, trained by the MMI method, that is described in in (Rigoll, 1997) and extended in (Neukirchen, 1998), are used as VQ to quantize the features into $Z = 4$ different streams of discrete labels. The codebook size in each stream is set to 200.

A simple system with models for 47 monophones and for the most prominent 33 function words (totally 394 states) yields a word error rate of 8.6%. A system that makes use of the more detailed (but untied) word internal triphone models (totally 6921 states) yields 12.2% word error. Hence HMM overfitting because of insufficient training data is a severe problem in this case. Traditional methods to overcome the effects of overfitting like interpolating between triphones and monophones (Bahl, 1983), data driven state-clustering and decision tree clustering yield error rates of 6.5%, 8.3% and 6.4%, respectively. It must be noted that in contrast to the usual training procedure in (Rigoll, 1996) no further smoothing methods are applied to the HMM emission probabilities here.

In a first series of experiments the untied triphone system is regulized by a quite simple mixture of $I = 394$ density components, i.e. the number of clusters in the penalty term is identical to the number of states in the monophone system. In this case the prototype probabilities are initialized by the emission probabilities of the monophone system; the mixture weights and the deviation parameters in the regulizer are set to be uniform, initially. In order to test the inluence of the tradeoff parameter $\nu$ it is set to 50, 10 and 2, respectively. The corresponding word error rates are 8.4%, 6.9% and 6.3%, respectively. In the case of large $\nu$s regularization degrades to a tying of triphone states to monophone states and the error rate tends towards the monophone system performance. For smaller $\nu$s there is a good tradeoff between data fitting and HMM smoothness yielding improved system performance. The initial prototype probability settings provided by the monophone system do not seem to be changed much by regulizer parameter estimation, since the system performance only changes slightly when the regulizer's parameter reestimation is not incorporated.

In preliminary experiments the regularization method is also used for speaker adaptation. A speaker-independent system trained on the Wall Street Journal (WSJ) database yields an error rate of 32.4% on the Nov. 93 S3_P0 test set with 10 different non-native speakers. The speaker-independent HMM emission probabilities are used to initialize the prototype probabilities of the regulizer. Then, speaker-dependent systems are built up for each speaker using only 40 fast enrollment sentences for training along with regularization ($\nu$ is set to 10). Now, the error rate drops to 25.7% what is better than the speaker adaptation method described in (Rottland, 1998) that yields 27.3% by a linear feature space transformation. In combination both methods achieve 23.0% word error.

# 6   Summary and Discussion

A method to avoid parameter overfitting in HMM systems by application of a regularization term that favor smooth and simple models has been presented here. The complexity

measure applied to the HMMs is based on a finite mixture of negative exponential distributions, that generates the state-dependent emission probabilities. This kind of regularization term can be interpreted as a soft state-tying, since it forces the HMM emission probabilities to form a finite set of clusters. The effect of regularization has been demonstrated on the RM task by improving overfitted triphone models. On a WSJ non-native speaker adaption task with limited training data, regularization outperforms feature space transformations.

Eqn. (4) may be also interpreted from a perspective of Bayesian inference: the term $\nu \cdot \Omega$ plays the role of setting a prior distribution on the HMM parameters to be estimated. Hence, the use of a mixture model for $\Omega$ is equivalent to using a special kind of prior in the framework of MAP estimation for HMMs (Gauvain, 1994).

## References

L.R. Bahl, F. Jelinek, L.R. Mercer, 'A Maximum Likelihood Approach to Continuous Speech Recognition', *IEEE Trans. Pattern Analysis and Machine Intelligence*, Vol. 5, No. 2 Mar. 1983, pp. 179–190.

L.R. Bahl, P.V. de Souza, P.S. Gopalakrishnan, D. Nahamoo, M.A. Picheny, (1991) Context dependent modeling of phones in continuous speech using decision trees. *Proc. DARPA speech and natural language processing workshop*, 264–270.

E.B. Baum, D. Haussler, (1989) What size net gives valid generalization? *Neural Computation*, 1:151–160.

Y. le Cun, J. Denker, S. Solla, R.E. Howard, L.D. Jackel, (1990) Optimal brain damage. *Advances in Neural Information Processing Systems 2*, San Mateo, CA, Morgan Kauffman.

J.L. Gauvain, C.H. Lee, (1994) Maximum a posteriori estimation for multivariate Gaussian mixture observations of markov chains. *IEEE Transaction Speech and Audio Proc.*, Vol. 2, 2:291–298.

K.J. Lang, A.H. Waibel, G.E. Hinton, (1990) A time-delay neural network architecture for isolated word recognition. *Neural Networks*, 3:23–43.

Ch. Neukirchen, D. Willett, S. Eickeler, S. Müller, (1998) Exploiting acoustic feature correlations by joint neural vector quantizer design in a discrete HMM system. *Proc. ICASSP'98*, 5-8.

S.J. Nowlan, G.E. Hinton, (1992) Simplifying neural networks by soft weight-sharing. *Neural Computation*, 4:473–493.

D.C. Plaut, S.J. Nowlan, G.E. Hinton, (1986) Experiments on learning by backpropagation. *technical report CMU-CS-86-126*, Carnegie-Mellon University, Pittsburgh, PA.

G. Rigoll, Ch. Neukirchen, J. Rottland, (1996) A new hybrid system based on MMI-neural networks for the RM speech recognition task. *Proc. ICASSP'96*, 865–868.

G. Rigoll, Ch. Neukirchen, (1997) A new approach to hybrid HMM/ANN speech recognition using mutual information neural networks. *Advances in Neural Information Processing Systems 9*, Cambridge, MA, MIT Press, 772-778.

J. Rottland, Ch. Neukirchen, G. Rigoll, (1998) Speaker adaptation for hybrid MMI-connectionist speech recognition systems. *Proc. ICASSP'98*, 465–468.

S.J. Young, (1992) The general use of tying in phoneme-based HMM speech recognizers. *Proc. ICASSP'92*, 569–572.

S.J. Young, P.C. Woodland (1993) The use of state tying in continuous speech recognition. *Proc. Eurospeech'93*, 2203–2206.
